# Forward Dynamics Modeling of Speech Motor Control Using Physiological Data

**Makoto Hirayama  Eric Vatikiotis-Bateson  Mitsuo Kawato**
ATR Auditory and Visual Perception Research Laboratories
2 - 2, Hikaridai, Seika-cho, Soraku-gun, Kyoto 619-02, JAPAN

**Michael I. Jordan**
Department of Brain and Cognitive Sciences
Massachusetts Institute of Technology
Cambridge, MA 02139

## Abstract

We propose a paradigm for modeling speech production based on neural networks. We focus on characteristics of the musculoskeletal system. Using real physiological data – articulator movements and EMG from muscle activity – a neural network learns the forward dynamics relating motor commands to muscles and the ensuing articulator behavior. After learning, simulated perturbations, were used to asses properties of the acquired model, such as natural frequency, damping, and interarticulator couplings. Finally, a cascade neural network is used to generate continuous motor commands from a sequence of discrete articulatory targets.

## 1  INTRODUCTION

A key problem in the formal study of human language is to understand the process by which linguistic intentions become speech. Speech production entails extraordinary coordination among diverse neurophysiological and anatomical structures from which unfolds through time a complex acoustic signal that conveys to listeners something of the speaker's intention. Analysis of the speech acoustics has not revealed the encoding of these intentions, generally conceived to be ordered strings of some basic unit, e.g., the phoneme. Nor has analysis of the articulatory system provided an answer, although recent pioneering work by Jordan (1986), Saltzman (1986), Laboissière (1990) and others

has brought us closer to an understanding of the articulatory-to-acoustic transform and has demonstrated the importance of modeling the articulatory system's temporal properties. However, these efforts have been limited to kinematic modeling because they have not had access to the neuromuscular activity of the articulatory structures.

In this study, we are using neural networks to model speech production. The principle steps of this endeavor are shown in Figure 1. In this paper, we focus on characteristics of the musculoskeletal system. Using real physiological data – articulator movements and EMG from muscle activity – a neural network learns the forward dynamics relating motor commands to muscles and the ensuing articulator behavior. After learning, a cascade neural network model (Kawato, Maeda, Uno, & Suzuki, 1990) is used to generate continuous motor commands.

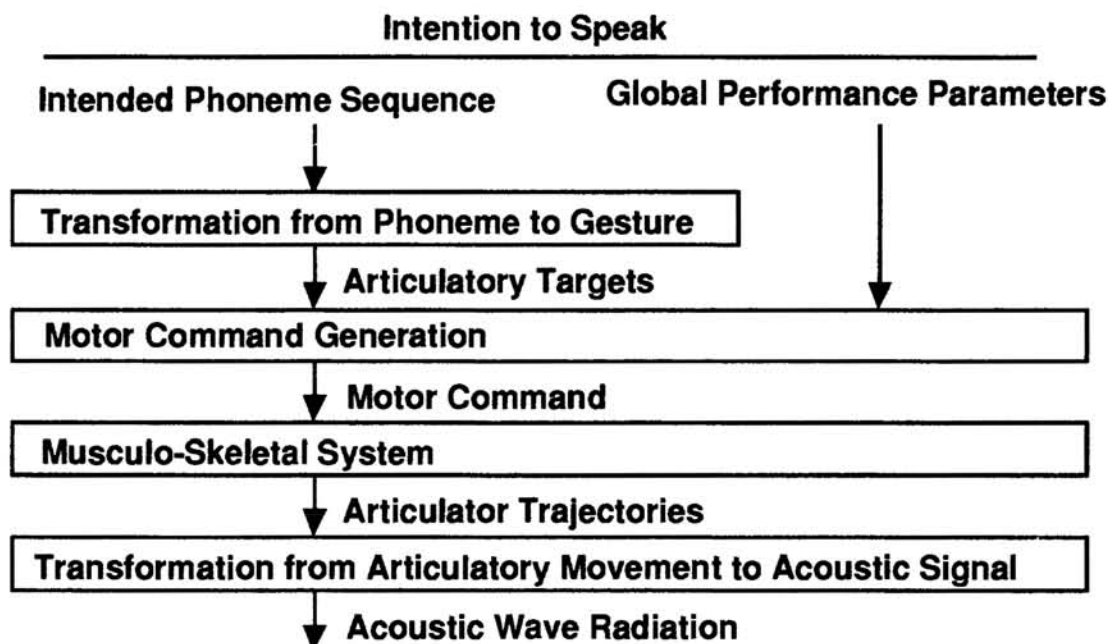

Figure 1: Forward Model of Speech Production

## 2  EXPERIMENT

Movement, EMG, and acoustic data were recorded for one speaker who produced reiterant versions of two sentences. Speaking rate was fast and the reiterant syllables were *ba, bo*. Figure 2 shows approximate marker positions for tracking positions of the jaw (horizontal and vertical) and lips (vertical only) and muscle insertion points for hooked-wire, bipolar EMG recording from four muscles: ABD (anterior belly of the digastric) for jaw lowering, OOI(orbicularis oris inferior) and MTL (mentalis) for lower lip raising and protrusion, and GGA (genioglossus anterior) for tongue tip lowering.

All movement and EMG (rectified and integrated) signals were digitized (12 bit) at 200 Hz and then numerically smoothed at 40 Hz. Position signals were differentiated to obtain velocity and then, after smoothing at 22 Hz, differentiated again to get acceleration. Figure 3 shows data for one reiterant utterance using *ba*.

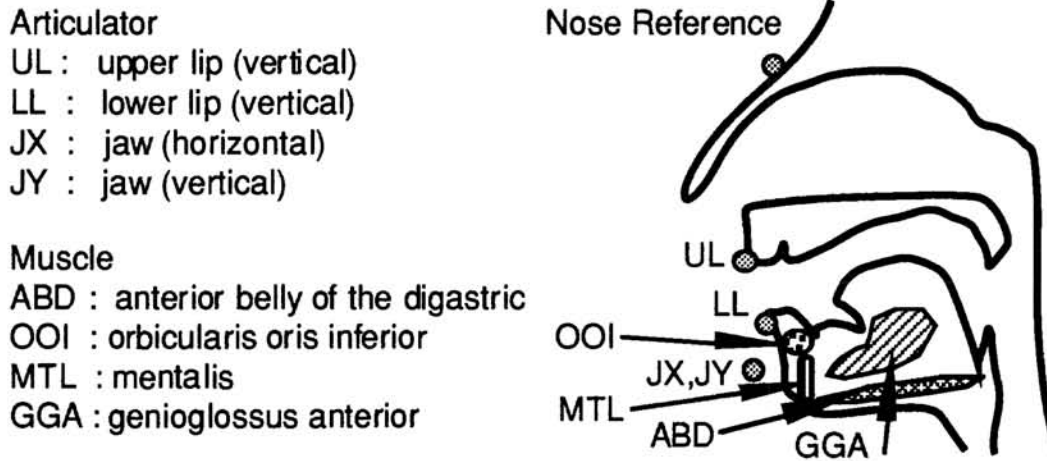

Articulator
UL :   upper lip (vertical)
LL  :   lower lip (vertical)
JX  :   jaw (horizontal)
JY  :   jaw (vertical)

Muscle
ABD :  anterior belly of the digastric
OOI  :  orbicularis oris inferior
MTL  : mentalis
GGA :  genioglossus anterior

Figure 2: Approximate Positions of Markers and Muscle Insertion
for Recording Movement and EMG

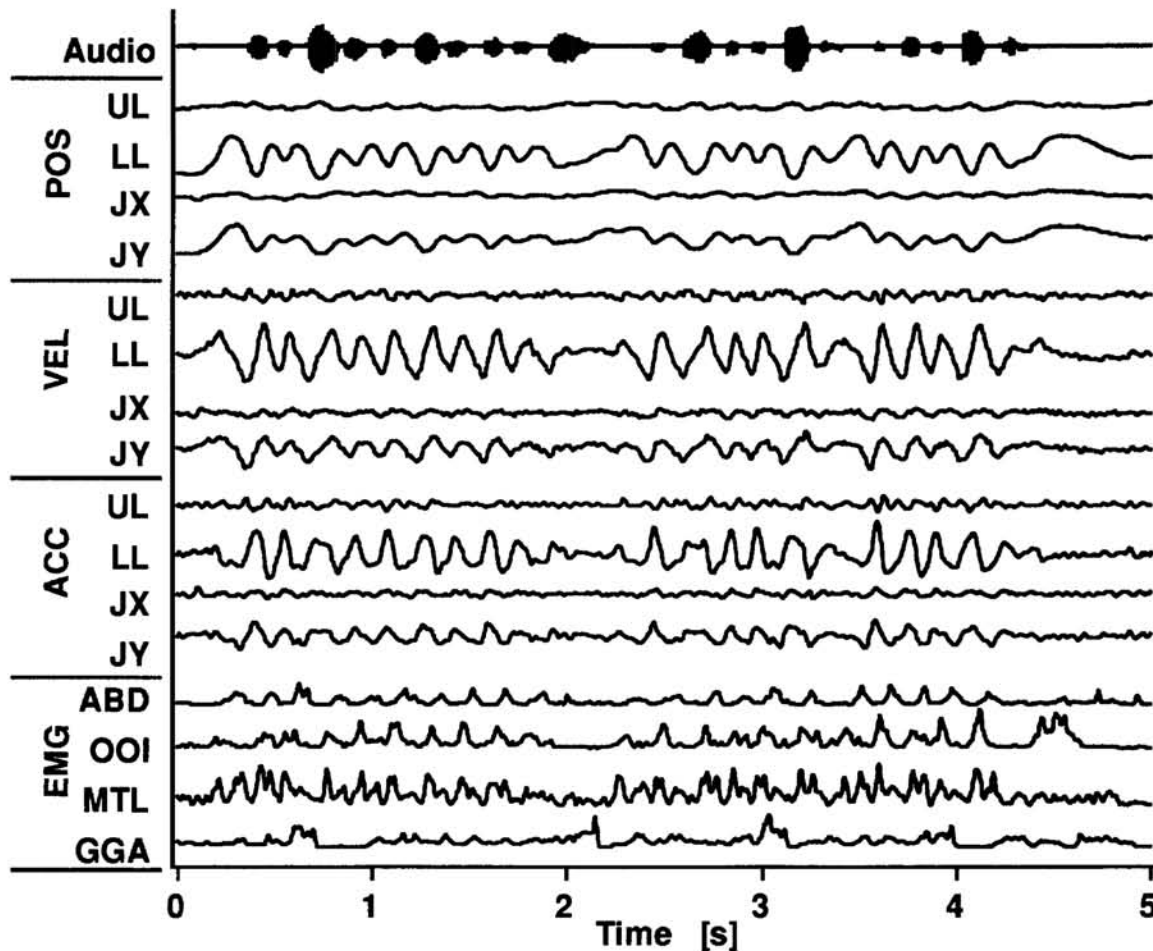

Figure 3: Time Series Representations for All Channels
of One Reiterant Rendition Using *ba*

## 3  FORWARD DYNAMICS MODELING OF THE MUSCULO-SKELETAL SYSTEM AND TRAJECTORY PREDICTION FROM MUSCLE EMG

The forward dynamics model (FDM) for *ba, bo* production was obtained using a three-layer perceptron with back propagation (Rumelhart, Hinton, & Williams, 1986). The network learns the correlations between position, velocity, EMG at time *t* and the changes of position and velocity for all articulators at the next time sample *t+1*.

After learning, the forward dynamics model is connected recurrently as shown in Figure 4. The network uses only the initial articulator position and velocity values and the continuous EMG "motor command" input to generate predicted trajectories. The FDM estimates the changes of position and velocity and sums them with position and velocity values of the previous sample *t* to obtain estimated values at the next sample *t+1*.

Figure 5 compares experimentally observed trajectories with trajectories predicted by this network. Spatiotemporal characteristics are very similar, e.g., amplitude, frequency, and phase, and demonstrate the generally good performance of the model. There is, however, a tendency towards negative offset in the predicted positions. There are two important limitations that reduce the current model's ability to compensate for position shifts in the test utterance. First, there is no specified equilibrium or rest position in articulator space, towards which articulators might tend in the absence of EMG activity. Second, the acquired FDM is based on limited EMG; at most there is correlated EMG for only one direction of motion per articulator. Addition of antagonist EMG and/or an estimate of equilibrium position in articulator or, eventually, task coordinates should increase the model's generalization capability.

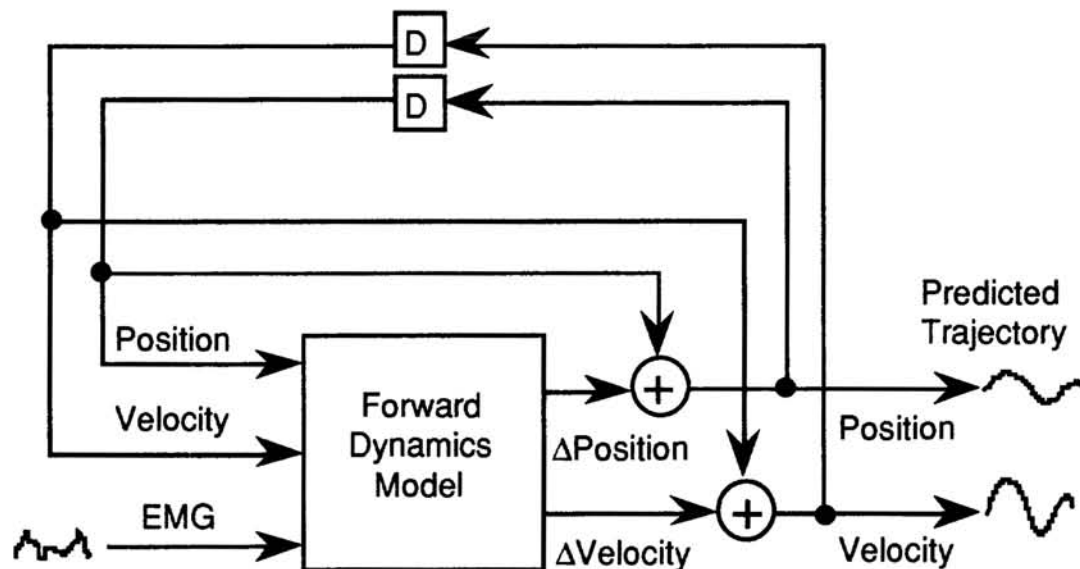

Figure 4: Recurrent Network for Trajectory Prediction from Muscle EMG

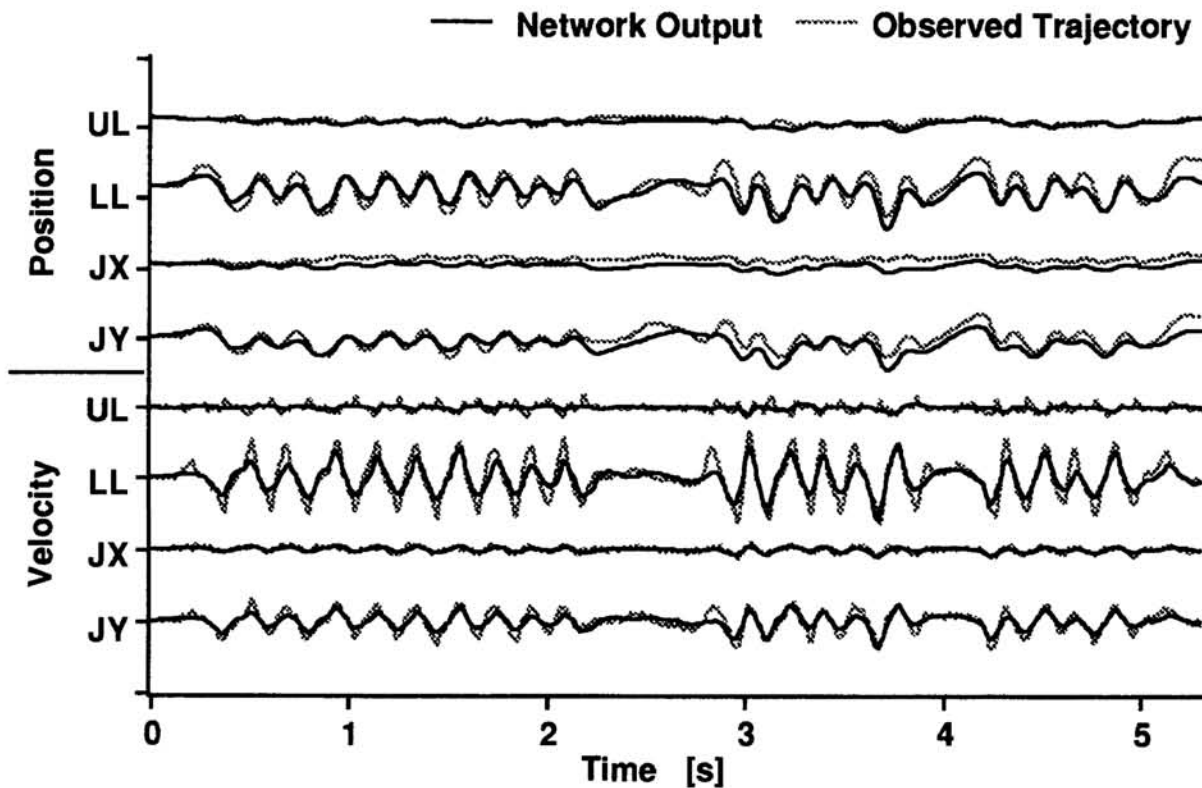

Figure 5: Experimentally Observed *vs*. Predicted Trajectories

## 4  ESTIMATION OF DYNAMIC PARAMETER

To investigate quantitative characteristics of the obtained forward dynamics model, the model system's response to two types of simulated perturbation were examined.

The first simulated perturbation confirmed that the model system indeed learned an appropriate nonlinear dynamics and affords a rough estimation of the its visco-elastic properties, such as natural frequency (1.0 Hz) and damping ratio (0.24). Simulated release of the lower lip at various distances from rest revealed underdamped though stable behavior, as shown in Figure 6a.

The second perturbation entailed observing articulator response to a step increase (50 % of full-scale) in EMG activity for each muscle. Figure 6b demonstrates that the learned relation between EMG input and articulator movement output is dynamical rather than kinematic because articulator responses are not instantaneous. Learned responses to each muscle's activation also show some interesting and reasonable (though not always correct) couplings between different articulators.

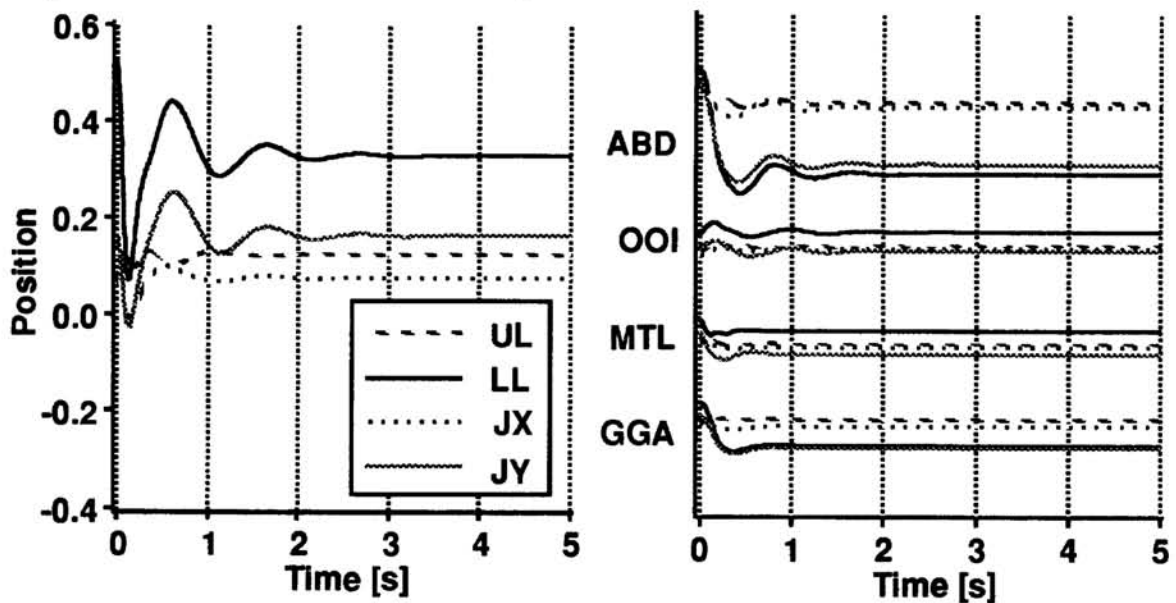

**a.** Release of Lower Lip
    from Rest Position + 0.2

**b.** Response of Step
    Increase (+0.5) in EMG

Figure 6: Visco-Elastic Property of the FDM Observed by Simulated Perturbations

## 5   MOTOR COMMAND GENERATION USING CASCADE NEURAL NETWORK MODEL

Observed articulator movements are smooth. Their smoothness is due partly to physical dynamic properties (inertia, viscosity). Furthermore, smoothness may be an attribute of the motor command itself, thereby resolving the ill-posed computational problem of generating continuous motor commands from a small number of discrete articulatory targets.

To test this, we incorporated a smoothness constraint on the motor command (rectified EMG, in this case), which is conceptually similar to previously proposed constraints on change of torque (Uno, Kawato, & Suzuki, 1989) and muscle-tension (Uno, Suzuki, & Kawato, 1989). Two articulatory target (via-point) constraints were specified spatially, one for consonant closure and the other for vowel opening, and assigned to each of the 21 consonant + vowel syllables. The alternating sequence of via-points was isochronous (temporally equidistant) except for initial, medial and final pauses. The cascade neural network (Figure 7) then generated smooth EMG and articulator trajectories whose spatiotemporal asymmetry approximated the prosodic patterning of the natural test utterances (Figure 8). Although this is only a preliminary implementation of via-point and smoothness constraints, the model's ability to generate trajectories of appropriate spatiotemporal complexity from a series of alternating via-point inputs is encouraging.

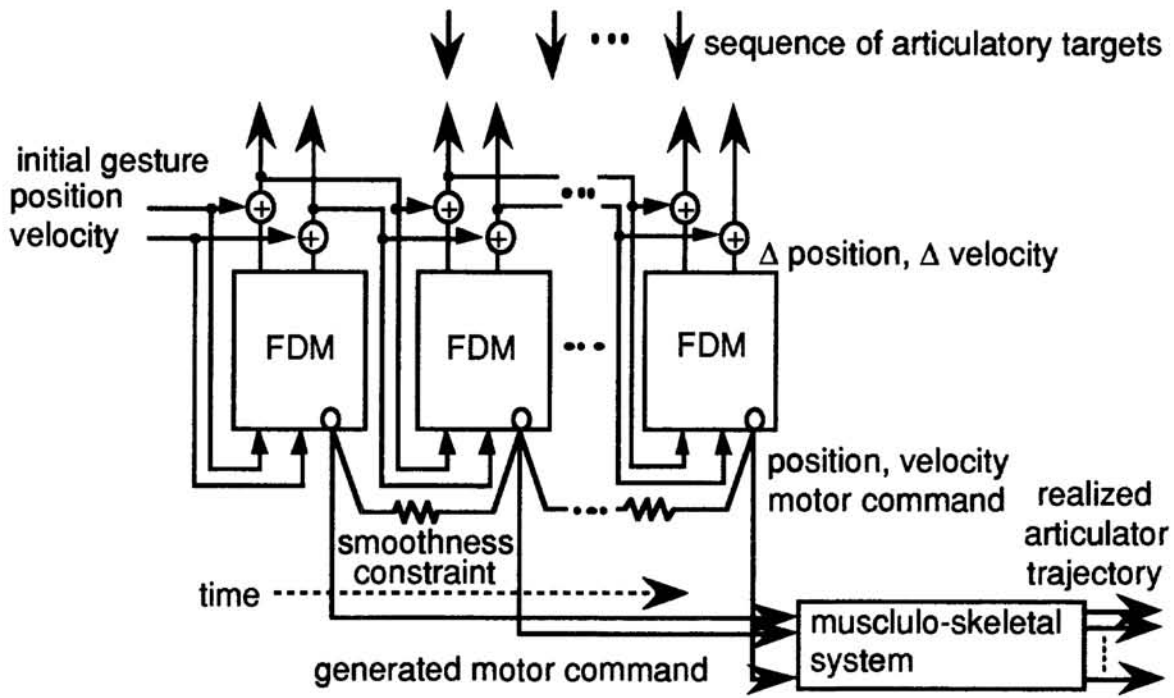

Figure 7: Cascade Neural Network Model for Motor Command Generation

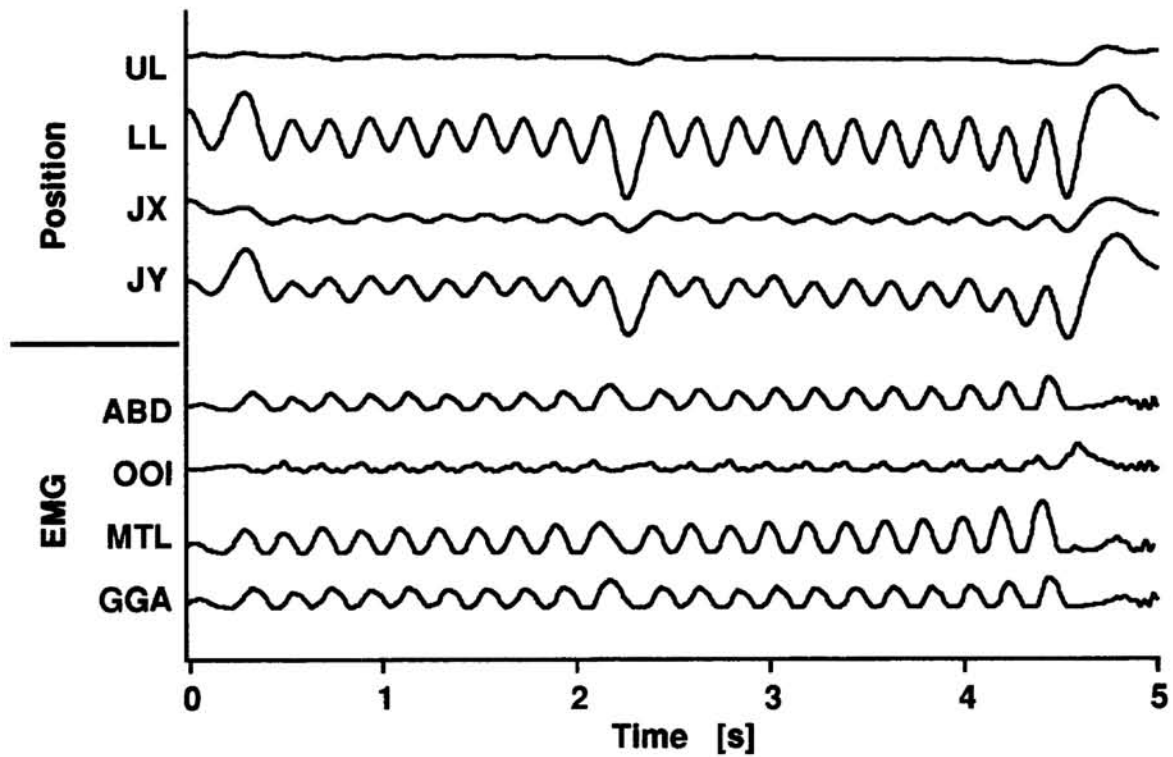

Figure 8: Generated Motor Command (EMG) with Trajectory
To Satisfy Articulatory Targets

## 6  CONCLUSION AND FUTURE WORK

Our intent here has been to provide a preliminary model of speech production based on the articulatory system's dynamical properties. We used real physiological data — EMG — to obtain the forward dynamics model of the articulators from a multilayer perceptron. After training, a recurrent network predicted articulator trajectories using the EMG signals as the motor command input. Simulated perturbations were used to examine the model system's response to isolated inputs and to assess its visco-elastic properties and interarticulator couplings. Then, we incorporated a reasonable smoothness criterion — minimum-motor-command-change — into a cascade neural network that generated realistic trajectories from a bead-like string of via-points.

We are now attempting to model various styles of real speech using data from more muscles and articulators such as the tongue. Also, the scope of the model is being expanded to incorporate global performance parameters for motor command generation, and the transformations from phoneme to articulatory gesture and from articulatory movement to acoustic signal.

Finally, a main goal of our work is to develop engineering applications for speech synthesis and recognition. Although our model is still preliminary, we believe resolving the difficulties posed by coarticulation, segmentation, prosody, and speaking style ultimately depends on understanding physiological and computational aspects of speech motor control.

### Acknowledgement

We thank Vincent Gracco and Kiyoshi Oshima for muscle insertions; Haskins Laboratories for use of their facilities (NIH grant DC-00121); Kiyoshi Honda, Philip Rubin, Elliot Saltzman and Yoh'ichi Toh'kura for insightful discussion; and Kazunari Nakane and Eiji Yodogawa for continuous encouragement. Further support was provided by HFSP grants to M. Kawato and M. I. Jordan.

# References

Jordan, M. I. (1986) Serial order: a parallel distributed processing approach, *ICS (Institute for Cognitive Science, University of California) Report*, **8604**.

Kawato, M., Maeda, M., Uno, Y. & Suzuki, R. (1990) Trajectory Formation of Arm Movement by Cascade Neural Network Model Based on Minimum Torque-change Criterion, *Biol. Cybern.* **62**, 275-288.

Laboissière, R., Schwarz, J. L. & Bailly, G. (1990) Motor Control for Speech Skills: a Connectionist Approach, *Proceeding of the 1990 Summer School*, Morgan Kaufmann Publishers, 319-327.

Rumelhart, D.E., Hinton, G.E. & Williams, R.J.(1986) Learning Internal Representation by Error Propagation, *Parallel Distributed Processing* Chap. 8, MIT Press.

Saltzman, E.L. (1986) Task dynamics coordination of the speech articulators: A preliminary model, *Experimental Brain Research*, Series 15, 129-144.

Uno, Y., Kawato, M., & Suzuki, R. (1989) Formation and Control of Optimal Trajectory in Human Multijoint Arm Movement, *Biol. Cybern.* **61**, 89-101.

Uno, Y., Suzuki, R. & Kawato, M. (1989) Minimum muscle-tension-change model which reproduces human arm movement, *Proceedings of the 4th symposium on Biological and Physiological Engineering*, 299-302, in Japanese.
